# Statistically Efficient Estimation Using Cortical Lateral Connections

**Alexandre Pouget**
alex@salk.edu

**Kechen Zhang**
zhang@salk.edu

## Abstract

Coarse codes are widely used throughout the brain to encode sensory and motor variables. Methods designed to interpret these codes, such as population vector analysis, are either inefficient, i.e., the variance of the estimate is much larger than the smallest possible variance, or biologically implausible, like maximum likelihood. Moreover, these methods attempt to compute a *scalar* or *vector* estimate of the encoded variable. Neurons are faced with a similar estimation problem. They must read out the responses of the presynaptic neurons, but, by contrast, they typically encode the variable with a further population code rather than as a scalar. We show how a non-linear recurrent network can be used to perform these estimation in an optimal way while keeping the estimate in a coarse code format. This work suggests that lateral connections in the cortex may be involved in cleaning up uncorrelated noise among neurons representing similar variables.

## 1 Introduction

Most sensory and motor variables in the brain are encoded with coarse codes, i.e., through the activity of large populations of neurons with broad tuning to the variables. For instance, direction of visual motion is believed to be encoded in visual area MT by the responses of a large number of cells with bell-shaped tuning, as illustrated in figure 1-A.

Neurophysiological recordings have shown that, in response to an object moving along a particular direction, the pattern of activity across such a population would look like a noisy hill of activity (figure 1-B). On the basis of this activity, $\vec{A}$, the best that can be done is to recover the conditional probability of the direction of motion, $\theta$, given the activity, $p(\theta|\vec{A})$. A slightly less ambitious goal is to come up with a good "guess", or estimate, $\hat{\theta}$, of the direction, $\theta$, given the activity. Because of the stochastic nature of the noise, the estimator is a random variable, i.e, for

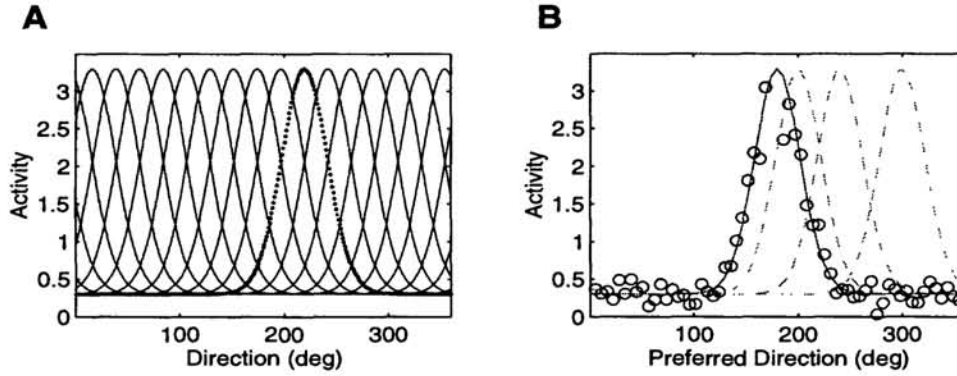

Figure 1: A- Tuning curves for 16 direction tuned neurons. B- Noisy pattern of activity (o) from 64 neurons when presented with a direction of 180°. The ML estimate is found by moving an "expected" hill of activity (dotted line) until the squared distance with the data is minimized (solid line)

the same image, $\hat{\theta}$ will vary from trial to trial. A good estimator should have the smallest possible variance across those trials because the variance determines how well two similar directions can be discriminated using this estimator. The Cramér-Rao bound provides an analytical lower bound for this variance given the noise in the system and the unit tuning curves [5] . Typically, computationally simple estimators, such as optimum linear estimator (OLE), are very inefficient; their variances are several times the bound. By contrast, Bayesian or maximum likelihood (ML) estimators (which are equivalent for the case under consideration in this paper) can reach this bound but require more complex calculations [5].

These decoding technics are valuable for a neurophysiologist interested in reading out the population code but they are not directly relevant for understanding how neural circuits perform estimation. In particular, they all provide the estimate in a format which is incompatible with what we know of sensory representations in the cortex. For example, cells in V4 are estimating orientation from the noisy responses of orientation tuned V1 cells, but, unlike ML or OLE which provide a scalar estimate, V4 neurons retain orientation in a coarse code format, as demonstrated by the fact that V4 cells are just as broadly tuned to orientation as V1 neurons.

Therefore, it seems that a theory of estimation in biological networks should have two critical characteristics: 1- it should preserve the estimate in a coarse code and 2- it should be efficient, i.e., the variance should be close to the Cramér-Rao bound. We explore in this paper various network architectures for performing estimations with coarse code using lateral connections. We start by briefly describing several classical estimators such as OLE or ML. Then, we consider linear and non-linear recurrent networks and compare their performances with the classical estimators.

## 2   Classical Methods

The simplest estimators are linear of the form $\hat{\theta}_{OLE} = \vec{W}^T \vec{A}$. Better performance can be obtained with a center of mass estimator (COM), $\hat{\theta}_{COM} = \sum_i \theta_i a_i / \sum_i a_i$; however, in the case of a periodic variable, such as direction of motion, the best one-shot method known is the complex estimator (COMP), $\hat{\theta}_{COMP} = phase(z)$ where $z = \sum_{k=1}^{N} a_k e^{i\theta_k}$ [5]. This estimator consists in fitting a cosine through the pattern of activity, like the one shown in figure 1-B, and using the phase of

**A**

**B**     Activity over Time

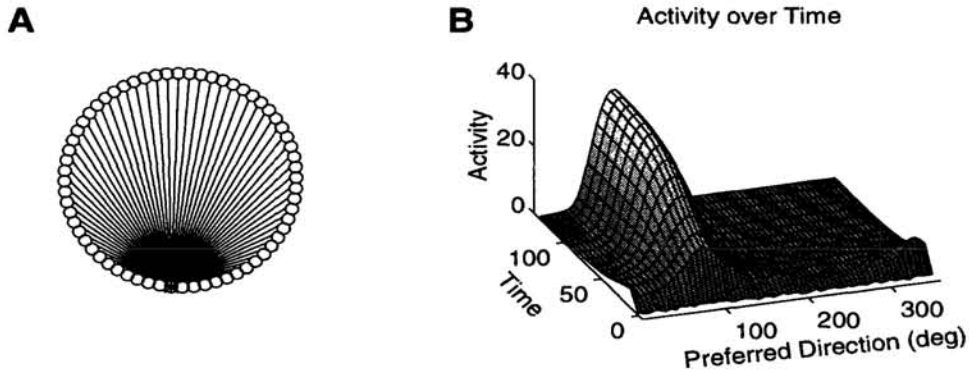

Figure 2: A- Circular network of 64 units. Only the connections originating from one unit are shown. B- Activity over time in the non-linear network when initialized with a random pattern at $t = 0$. The activity of the units are plotted as a function of their position along the circle which is equivalent to their preferred direction of motion with appropriate choice of weights.

the best cosine fit as the estimate of direction. This method is suboptimal if the data were not generated by cosine tuning functions as in the case illustrated in figure 1-A. It is possible to obtain optimum performance by fitting the curve that was actually used to generate the data, i.e, the actual tuning curves of the units. A maximum likelihood estimate, defined as being the direction maximizing $p(\vec{A}|\theta)$, involves exactly this type of curve fitting, a process illustrated in figure 1-B [5]. The estimate is computed by finding first the "expected" hill– the hill that would be obtained in a noise free system– minimizing the distance with the data. In the case of gaussian noise, the appropriate distance measure to minimize is the euclidian squared distance. The final position of the peak of the hill corresponds to the maximum likelihood estimate, $\hat{\theta}_{ML}$.

## 3   Recurrent Networks

Consider a circular network of 64 units fully connected like the one depicted in figure 2-A. With an appropriate choice of weights and activation function, this network will develop a hill-shaped pattern of activity in response to a transient input as illustrated in figure 2-B. If we initialize this networks with activity patterns $\vec{A} = \{a_i\}$ corresponding to the responses of 64 direction tuned units (figure 1), we can use the final position of the hill across the neuronal array after relaxation as an estimate of the direction, $\theta$. The variance of this estimator will depend on the exact choice of activation function and weights.

### 3.1   Linear Network

We first consider a network of 64 units whose dynamics is governed by the following difference equation:

$$o_i(t + \delta t) = o_i(t) + \delta t \left( -o_i(t) + \sum_{j=1}^{N} w_{ij} o_j(t) \right) \qquad (1)$$

The dynamics of such networks is well understood [3]. If each unit receives the same weight vector $\vec{w}$, then the weight matrix $W$ is symmetric. In this case, the

network dynamics amplifies or suppresses the Fourier component of the initial input pattern, $\{a_i\}$, independently by a factors equal to the corresponding component of the Fourier transform, $\vec{\tilde{w}}$, of $\vec{w}$. For example, if the first component of $\vec{\tilde{w}}$ is more than one (resp. less than one) the first Fourier component of the initial pattern of activity will be amplified (resp. suppressed).

Thus, we can choose W such that the network amplifies selectively the first Fourier component of the data while suppressing the others. The network would be unstable but if we stop after a large, yet fixed, number of iterations, the activity pattern would look like a cosine function of direction with a phase corresponding to the phase of the first Fourier components of the data. In other words, the network would end up fitting a cosine function in the data which is equivalent to the COMP method described above. A network for orientation selectivity proposed by Ben-Yishai et al [1] is closely related to this linear network.

Although this method keeps the estimate in a coarse code format, it suffers two problems: it is unclear how it could be extended to non periodic variables, such as disparity, and it is suboptimal since it is equivalent to the COMP estimator.

## 3.2   Non-Linear Network

We consider next a network of 64 units fully connected whose dynamics is governed by the following difference equations:

$$o_i(t) = g(u_i(t)) = 6.3 \left( \log \left( 1 + e^{5+10u_i(t)} \right) \right)^{0.8} \tag{2}$$

$$u_i(t + \delta t) = u_i(t) + \delta t \left( -u_i(t) + \sum_{j=1}^{N} w_{ij} o_j(t) \right) \tag{3}$$

Zhang (1996) has demonstrated that with appropriate symmetric weights, $\{w_{ij}\}$, this network develops a stable hill of activity in response to an arbitrary transient input pattern $\{I_i\}$(figure 2-B). The shape of the hill is fully specified by the weights and activation function whereas, by contrast, the final position of the hill on the neuronal array depends only on the initial input. Therefore, like ML, the network fits an "expected" function through the data. We first present a set of simulations in which we investigated whether ML and the network place the hill at the same location.

**Methods:** The simulations consisted estimating the value of the direction of a moving bar based on the activity, $\vec{A} = \{a_i\}$, of 64 input units with hill-shaped tuning to direction corrupted by noise. We used circular normal functions like the ones showed in figure 1-A to model the mean activities, $f_i(\theta)$:

$$f_i(\theta) = 3 \exp(7(\cos(\theta - \theta_i) - 1)) + 0.3 \tag{4}$$

The value 0.3 corresponds to the mean spontaneous activity of each unit. The peak, $\theta_i$, of the circular normal functions were uniformly spread over the interval $[0°, 360°]$. The activities, $\{a_i\}$, depended on the noise distribution. We used two types of noise, normally distributed with fixed variance, $\sigma_n^2 = 1$ and Poisson distributed:

$$P(a_i = a|\theta) = \frac{1}{\sqrt{2\pi\sigma_n^2}} \exp\left( -\frac{(a - f_i(\theta))^2}{2\sigma_n^2} \right), \quad P(a_i = k|\theta) = \frac{f_i(\theta)^k e^{-f_i(\theta)}}{k!} \tag{5}$$

Our results compare the standard deviation of four estimators, OLE, COM, COMP and ML to the non-linear recurrent network (RN) with *transient* inputs (the input patterns are shown on the first iteration only). In the case of ML, we used the

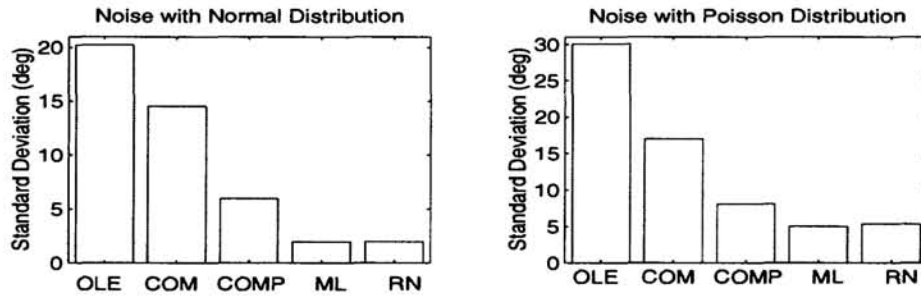

Figure 3: Histogram of the standard deviations of the estimate for all five methods

Cramér-Rao bound to compute the standard deviation as described in Seung and Sompolinsky (1993). The weights in the recurrent network were chosen such that the final pattern of activity in the network have a profile very similar to the tuning function $f_i(\theta)$.

**Results:** Since the preferred direction of two consecutive units in the network are more than 5° apart, we first wonder whether RN estimates exhibit a bias— a difference between the mean estimate and the true direction— in particular for directions between the peaks of two consecutive units. Our simulations showed no significant bias for any of the orientations tested (not shown). Next, we compared standard deviations of the estimates for all five methods and for the two types of noise. The RN method was found to outperform the OLE, COM and COMP estimators in both cases and to match the Cramér-Rao bound for gaussian noise (figure 3) as suggested by our analysis. For noise with Poisson distribution, the standard deviation for RN was only 0.344° above ML (figure 3).

We also estimated numerically $-\partial\hat{\theta}_{RN}/\partial a_i|_{\theta=170°}$, the derivative of the RN estimate with respect to the *initial* activity of each of 64 units for an orientation of 170°. This derivative in the case of ML matches closely the derivative of the cell tuning curve, $f'(\theta)$. In other words, in ML, units contribute to the estimate according to the amplitude of the derivative of the tuning curve. As shown in figure the same is true for RN, $-\partial\hat{\theta}_{RN}/\partial a_i|_{\theta=170°}$ matches closely the derivative of the units tuning curves. In contrast, the same derivatives for the COMP estimate, (dotted line), or the COM estimate, (dash-dotted line), do not match the profile of $f'(\theta)$. In particular, units with preferred direction far away from 170°, i.e. units whose activity is just noise, end up contributing to the final estimate, hindering the performance of the estimator.

We also looked at the standard deviation of the RN as a function of time, i.e., the number of iterations. Reaching a stable state can take up to several hundred iterations which could make the RN method too slow for any practical purpose. We found however that the standard deviation decreases very rapidly over the first 5-6 iterations and reaches asymptotic values after around 20 iterations (figure 4-B). Therefore, there is no need to wait for a perfectly stable pattern of activity to obtain minimum standard deviation.

**Analysis:** One way to determine which factors control the final position of the hill is to find a function, called a Lyapunov function, which is minimized over time by the network dynamics. Cohen and Grossberg (1983) have shown that network characterized by the dynamical equation above and in which the input pattern $\{sI_i\}$

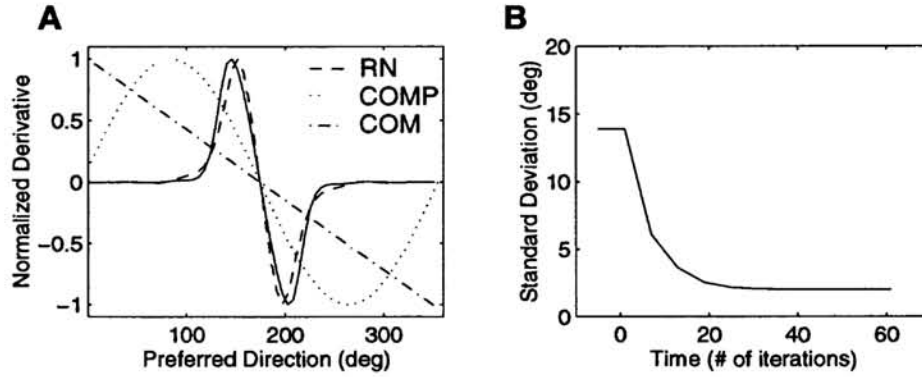

Figure 4: A- Comparison of $g'(\theta)$ (solid line), $-\partial\hat{\theta}/\partial a_i|_{\theta=170°}$ for RN, COMP and COM. All functions have been normalized to one. B- Standard deviation as a function of the number of iterations for RN.

is clamped, minimizes a Lyapunov function of the form:

$$L = -\frac{1}{2}\sum_{i,j} w_{ij}g(u_i)g(u_j) + \sum_i \int_0^{u_i} zg'(z)\,dz - s\sum_i I_i g(u_i). \qquad (6)$$

The last term is the dot product between the input pattern, $\{sI_i\}$, and the current activity pattern, $\{g(u_i)\}$, on the neuronal array. Here $s$ is simply a scaling factor for the input pattern. The dynamics of the network will therefore tend to minimize $-\sum_i I_i g(u_i)$, or equivalently, to maximize the overlap between the stable pattern and the input pattern. The other terms however are also dependent on $I_i$ because the shape of the final stable activity profile depends on the input pattern. Therefore the network will settle into a compromise between maximizing overlap and getting the right profile given the clamped input.

We can show however that, for small input (i.e., as the scaling factor $s \to 0$), the dominant term in the Lyapunov function is the dot product. To see this, we consider the Taylor expansion of Lyapunov function $L$ with respect to $s$. First, let $\{U_i\}$ denote the profile of the stable activity $\{u_i\}$ in the limit of zero input ($s \to 0$), and then write the corresponding value of the Lyapunov function at zero input as $L_0$. Now keeping only the first-order terms of $s$ in the Taylor expansion, we obtain:

$$L \approx L_0 - s\sum_i I_i g(U_i). \qquad (7)$$

This means that the dot product is the only first order term of $s$, and disturbances to the shape of the final activity profile contribute only to higher order terms of $s$, which are negligible when $s$ is small. Notice that in the limit of zero input, the shape of the activity profile $\{U_i\}$ is fixed, and the only thing unknown is its peak position. Because $L_0$ is a constant, the global minimum of the Lyapunov function here should correspond to a peak position which maximizes the dot product. The difference between $u_i$ and $U_i$ is negligible for sufficiently small input because, by definition, $u_i \to U_i$ as $s \to 0$. Consequently, for small input, the network will converge to a solution maximizing primarily $\sum_i I_i g(u_i)$, which is mathematically equivalent to minimizing the square distance between the input and the output pattern.

Therefore, if we use an activity pattern, $\vec{A} = \{a_i\}$, as the input to this network, the stable hill should have its peak at a position very close to the direction corre-

sponding to the maximum likelihood estimate (under the assumption of gaussian noise), provided the network is not attracted into a local minimum of the Liapunov function. This result is valid when using a small *clamped* input but our simulations show that a *transient* input is sufficient to reach the Cramér-Rao bound.

## 4 Discussion

Our results demonstrate that it is possible to perform efficient unbiased estimation with coarse coding using a neurally plausible architecture. Our model relies on lateral connections to implement a prior expectation on the profile of the activity patterns. As a consequence, units determine their activation according to their own input and the activity of their neighbors. This approach shows that one of the advantages of coarse code is to provide a representation which simplifies the problem of cleaning up uncorrelated noise within a neuronal population.

Unlike OLE, COM and COMP, the RN estimate is not the result of a voting process in which units vote from their preferred direction, $\theta_i$. Instead, units turn out to contribute according to the derivatives of their tuning curves, $f_i'(\theta)$, as in the case of ML. This feature allows the network to ignore background noise, that is to say, responses due to other factors beside the variable of interest. This property also predicts that discrimination of directions around the vertical (90°) would be most affected by shutting off the units tuned at 60° and 120°. This prediction is consistent with psychophysical experiments showing that discrimination around the vertical in human is affected by prior adaptation to orientations displaced from the vertical by ±30° [4].

Our approach can be readily extended to any other periodic sensory or motor variables. For non periodic variables such as the disparity of a line in an image, our network needs to be adapted since it currently relies on circular symmetrical weights. Simply unfolding the network will be sufficient to deal with values around the center of the interval under consideration, but more work is needed to deal with boundary values. We can also generalize this approach to arbitrary mapping between two coarse codes for variables $x$ and $y$ where $y$ is a function of $x$. Indeed, a coarse code for $x$ provides a set of radial basis functions of $x$ which can be subsequently used to approximate arbitrary functions. It is even conceivable to use a similar approach for one-to-many mappings, a common situation in vision or robotics, by adapting our network such that several hills can coexist simultaneously.

## Footnotes

[0]AP is at the Institute for Computational and Cognitive Sciences, Georgetown University, Washington, DC 20007 and KZ is at The Salk Institute, La Jolla, CA 92037 . This work was funded by McDonnell-Pew and Howard Hughes Medical Institute.

## References

[1] R. Ben-Yishai, R. L. Bar-Or, and H. Sompolinsky. *Proc. Natl. Acad. Sci. USA*, 92:3844–3848, 1995.

[2] M. Cohen and S. Grossberg. *IEEE Trans. SMC*, 13:815–826, 1983.

[3] M. Hirsch and S. Smale. *Differential equations, dynamical systems and linear algebra*. Academic Press, New York, 1974.

[4] D. M. Regan and K. I. Beverley. *J. Opt. Soc. Am.*, 2:147–155, 1985.

[5] H. S. Seung and H. Sompolinsky. *Proc. Natl. Acad. Sci. USA*, 90:10749–10753, 1993.
